# Going Metric: Denoising Pairwise Data

**Volker Roth**
Informatik III, University of Bonn
Roemerstr 164, 53117 Bonn, Germany
roth@cs.uni-bonn.de

**Julian Laub**
Fraunhofer FIRST.IDA
Kekulestr. 7, 12489 Berlin, Germany
jlaub@first.fhg.de

**Joachim M. Buhmann**
Informatik III, University of Bonn
Roemerstr 164, 53117 Bonn, Germany
jb@cs.uni-bonn.de

**Klaus-Robert Müller**
Fraunhofer FIRST.IDA,
12489 Berlin, Germany,
University of Potsdam,
14482 Potsdam, Germany
klaus@first.fhg.de

## Abstract

Pairwise data in empirical sciences typically violate metricity, either due to noise or due to fallible estimates, and therefore are hard to analyze by conventional machine learning technology. In this paper we therefore study ways to work around this problem. First, we present an alternative embedding to multi-dimensional scaling (MDS) that allows us to apply a variety of classical machine learning and signal processing algorithms. The class of pairwise grouping algorithms which share the shift-invariance property is statistically invariant under this embedding procedure, leading to identical assignments of objects to clusters. Based on this new vectorial representation, denoising methods are applied in a second step. Both steps provide a theoretically well controlled setup to translate from pairwise data to the respective denoised metric representation. We demonstrate the practical usefulness of our theoretical reasoning by discovering structure in protein sequence data bases, visibly improving performance upon existing automatic methods.

## 1 Introduction

Unsupervised grouping or *clustering* aims at extracting hidden structure from data (see e.g. [5]). However, for several major applications, e.g. bioinformatics or imaging, the data is solely available as scores of pairwise comparisons. Pairwise data is in no natural way related to the common viewpoint of objects lying in some "well behaved" space like a vector space. Particularly, pairwise data may violate the triangular inequality. Two cases should be distinguished: (i) The triangle inequality might not be satisfied as a result of noisy measurements (for instance using string alignment algorithms in DNA analysis). (ii) The violation might be an intrinsic feature of the data. This case, for instance, applies to datasets based upon some human judgment, e.g. "$X$ likes $Y$, $Y$ likes $Z \not\Rightarrow X$ likes $Z$".

Such violations preclude the use of well established machine learning methods, which typically have been formulated for metric data only. This paper proposes an algorithm to metricize and subsequently denoise pairwise data. It uses the so-called constant shift embedding (cf. [14]) for metrization, then constructs a positive semidefinite matrix which can in sequel be used for denoising and clustering purposes. Regarding data-mining or clustering purposes, the most outstanding difference to classical MDS is the following: for the class of pairwise clustering cost functions sharing the shift-invariance property[1] the metrization step is *loss-free* in the sense that the optimal assignments of objects to clusters remain unchanged.

The next section introduces techniques for metrization, denoising and clustering pairwise data. This is followed by a section illustrating our methods for real world data such as bacterial *GyrB* amino acid sequences and sequences from the ProDom data base and a brief discussion.

## 2 Proximity-based clustering and denoising

One of the most popular methods for grouping vectorial data is $k$-means clustering (see e.g. [1][5]). It derives a set of $k$ prototype vectors which quantize the data set with minimal quantization error.

Partitioning proximity data is considered a much harder problem, since the inherent structure of $n$ samples is hidden in $n^2$ pairwise relations. The pairwise proximities can violate the requirements of a distance measure, i.e. they may be non-symmetric and negative, and the triangular inequality does not necessarily hold. Thus, a loss-free embedding into a vector space is not possible, so that grouping problems of this kind cannot be directly transformed into a vectorial representation by means of classical embedding strategies such as multi-dimensional scaling (MDS [4]). Moreover clustering the MDS embedded data-vectors in general yields partitionings *different* from those obtained by directly solving the pairwise problem, since embedding constraints might be in conflict with the clustering goal.

Let us start from a pairwise clustering loss function (see [12]) that combines the properties of additivity, scale- and shift invariance, and statistical robustness

$$H^{\mathrm{pc}} = \sum_{\nu=1}^{k} \frac{\sum_{i=1}^{n} \sum_{j=1}^{n} M_{i\nu} M_{j\nu} D_{ij}}{\sum_{l=1}^{n} M_{l\nu}}, \tag{1}$$

where the data are characterized by the matrix of pairwise dissimilarities $D_{ij}$. The assignments of objects to clusters are encoded in the binary stochastic matrix $M \in \{0,1\}^{n \times k} : \sum_{\nu=1}^{k} M_{i\nu} = 1$. For such cost functions it can be shown [14] that there always exists a set of vectorial data representations—the constant shift embeddings—such that the grouping problem can be equivalently restated in terms of Euclidian distances between these vectors. In order to handle non-symmetric dissimilarities, it should be noticed that $H^{\mathrm{pc}}$ is also invariant under symmetrizing transformations: $D_{ij} \leftarrow 1/2(D_{ij} + D_{ji})$. In the following we will thus restrict ourselves to the case of symmetric dissimilarity matrices.

**Theorem 2.1.** *[14] Given an arbitrary (possibly non-metric) $(n \times n)$ dissimilarity matrix $D$ with zero self-dissimilarities, there exists a transformed matrix $\tilde{D}$ such that*
*(i) the matrix $\tilde{D}$ can be interpreted as a matrix of squared Euclidian distances*

between a set of vectors $\{x_i\}_{i=1}^n$. $\tilde{D}$ *is derived from* $D$ *by both symmetrizing and applying the constant shift embedding trick;*
*(ii) the original pairwise clustering problem is equivalent to a k-means problem in this vector space, in the sense that the optimal assignments of objects to clusters* $\{\hat{M}_{i\nu}\}$ *are* identical *in both problems.*

A re-formulation of pairwise clustering as a $k$-means problem is clearly advantageous: (i) the availability of prototype vectors defines a generic rule for using the learned partitioning in a predictive sense, (ii) we can apply standard noise- and dimensionality-reduction methods in order to both stabilize the estimation procedure and to speed up the grouping itself.

**Constant shift embedding** Let $D = (D_{ij}) \in \mathbb{R}^{n \times n}$ be the matrix of pairwise *squared* dissimilarities between $n$ objects. For a generic noisy dataset $\sqrt{D_{ij}} \nleq \sqrt{D_{ik}} + \sqrt{D_{kj}}$ so that $\sqrt{D}$ is non metric. Since $\sqrt{\cdot}$ is monotonically increasing, $\exists\, D_o$ such that $\sqrt{D_{ij} + D_o} \leqslant \sqrt{D_{ik} + D_o} + \sqrt{D_{kj} + D_o} \quad \forall\, i,j,k = 1,2\ldots n$. Let

$$\tilde{D} = D + D_o(ee^\top - I_n) \tag{2}$$

where $e = (1,1,\ldots 1)^\top$ is a $n$-dimensional column-vector and $I_n$ the identity matrix. This corresponds to a constant additive shift $\tilde{D}_{ij} = D_{ij} + D_o$ for all $i \neq j$. We look for the minimal constant shift $D_o$ such that $\tilde{D}$ satisfy the triangle inequality. In order to make the main result clear, we first need to introduce the notion of a *centralized matrix*. Let $P$ be an arbitrary matrix and let $Q = I - \frac{1}{n}ee^\top$. $Q$ is the projection matrix on the orthogonal complement of $e$. Define the *centralized* $P$ by:

$$P^c = QPQ. \tag{3}$$

Let $D$ be fixed and let us decompose $D$ as follows:

$$D_{ij} = S_{ii} + S_{jj} - 2S_{ij}. \tag{4}$$

This decomposition is motivated by the fact that if $D$ is a squared Euclidian distance between the vectorial data $x_i$, then $D_{ij} = \|x_i - x_j\|^2 = \|x_i\|^2 + \|x_j\|^2 - 2x_i^\top x_j$.
It follows from equation (4) that a constant off-diagonal shift on $D$ corresponds to a constant shift on the diagonal of $S$. $S$ is not fixed by the choice of $D$, since we may always change its diagonal elements, yet recover the same $D$. That is, any matrix of the form $(S_{ij} + 1/2\Delta S_i + 1/2\Delta S_j)$ gives the same distance $D$ as $S$ for arbitrary $\Delta S_i$'s. By simple algebra it can be shown that $S^c = -\frac{1}{2}D^c$, *i.e.* $S^c$ is unique. Furthermore $D$ derives from a squared Euclidian distance if and only if $S^c$ is positive semi-definite [14]. Let $\tilde{S}^c = S^c - \lambda_n(S^c)I_n$, where $\lambda_n(\cdot)$ is the minimal eigenvalue of its argument. Then $\tilde{S}^c$ is positive semi-definite [14]. These are the main ingredients for proving the following:

**Theorem 2.2 (Minimal $D_o$).** *[4].* $D_o = -2\lambda_n(S^c)$ *is the minimal constant such that* $\tilde{D} = D + D_o(ee^\top - I_n)$ *derive from squared Euclidian distance.*

All proofs can be found in [14]. We have thus shown that applying large enough additive shifts to the off-diagonal elements of $D$ results in a matrix $\tilde{S}^c$ that is positive semi-definite, and can thus be interpreted as a Gram matrix. This means, that in some $(n-1)$-dimensional Euclidian space there exists a vector representation of the objects, summarized in the "design" matrix $X$ (the rows of $X$ are the feature vectors), such that $\tilde{S}^c = XX^\top$.
For the pairwise clustering cost function the optimal assignments of objects to clusters are invariant under the constant-shift embedding procedure, according to

theorem 2.1. Hence, the grouping problem can be re-formulated as optimizing the classical $k$-means criterion in the embedding space.

In many applications, however, it is advantageous not to cluster in the full space but to insert some dimension reduction step, that serves the purpose of increasing efficiency and noise reduction. While it is unclear how to denoise for the original pairwise object representations while respecting additivity, scale- and shift invariance, and statistical robustness properties of the clustering criterion, we can easily apply kernel PCA [16] to $\tilde{S}^c$ after the constant-shift embedding.

**Denoising of pairwise data by Constant Shift Embedding** For denoising we construct $\tilde{D}$ which derives from "real" points in a vector space, i.e. $\tilde{S}^c$ is positive semi-definite. In a first step, we briefly describe, how these real points can be recovered by loss-free kernel PCA [16]:

(i) Calculate the centralized kernel matrix $S^c = -\frac{1}{2}QDQ$ .

(ii) Decompose $S^c = V\Lambda V^\top$ where $V = (v_1, \ldots v_n)$ with eigenvectors $v_i$'s and $\Lambda = \mathrm{diag}(\lambda_1, \ldots \lambda_n)$ with eigenvalues $\lambda_1 \geqslant \cdots \geqslant \lambda_p > \lambda_{p+1} = 0 \geqslant \lambda_{p+2} \geqslant \cdots \geqslant \lambda_n$.

(iii) Calculate the $n \times (n-2)$ mapping matrix $X^*_{n-2} = V^*_{n-2}(\Lambda^*_{n-2})^{1/2}$, where $V^*_{n-2} = (v_1, \ldots v_p, v_{p+2}, \ldots v_{n-1})$ and $\Lambda^*_{n-2} = \mathrm{diag}(\lambda_1 - \lambda_n, \ldots \lambda_p - \lambda_n, \lambda_{p+2} - \lambda_n, \ldots \lambda_{n-1} - \lambda_n)$ (these are the constantly shifted eigenvalues).

The rows of $X^*_{n-2}$ contain the vectors $\{x^*_i\}$ $(i = 1, 2 \ldots n)$ in $n-2$ dimensional space, whose mutual distances are given by $\tilde{D}$. When focusing on noise reduction, however, we are rather interested in some approximative reconstructions of the "real" vectors. In the PCA framework, one usually discards the directions which correspond to small eigenvalues as noise (c.f. [9]). We can thus obtain a representation in a space of reduced dimension (with the well-defined error of PCA reconstruction) when choosing $t < n-2$ in step (iii) of the above algorithm:

$$X^*_t = V^*_t(\Lambda^*_t)^{1/2},$$

where $V^*_t$ consists of the first $t$ column vectors of $V^*_{n-2}$ and $\Lambda^*_t$ is the top $t \times t$ submatrix of $\Lambda^*_{n-2}$. The vectors in $\mathbb{R}^t$ then differ the least from the vectors in $\mathbb{R}^{n-2}$ in the sense of a quadratic error.

The advantages of this method in comparison to directly applying classical scaling via MDS are: (i) $t$ can be larger than the number $p$ of positive eigenvalues, (ii) the embedded vectors are the best least squares error approximation to the optimal vectors which preserve the grouping structure.

It should be noticed, however, that given the exactly reconstructed vectors in $\mathbb{R}^{n-2}$ found by loss-free kernel PCA, we could have also applied any other standard methods for dimensionality reduction or visualization, such as *projection pursuit* [6], *local linear embedding* (LLE) [15], *Isomap* [17] or *Self-organizing maps* [8].

## 3 Application on protein sequences

### 3.1 Bacterial *GyrB* amino acid sequences

We first illustrate our denoising technique on the gyrase subunit B. The dataset consists of 84 amino acid sequences from five genera in *Actinobacteria*: 1: *Corynebacterium*, 2: *Mycobacterium*, 3: *Gordonia*, 4: *Nocardia* and 5: *Rhodococcus*. A detailed description can be found in [7]. This dataset was used in [18] for illustration of marginalized kernels. The authors hinted at the possibility of computing the distance matrix by using BLAST scores [2], noting, however, that these scores could not be converted into positive semidefinite kernels.

In our experiment, the sequences have been aligned by the Smith-Waterman algorithm [11] which yields pairwise alignment scores. Using constant shift embedding a *positive semidefinite* kernel is obtained, leaving the cluster assignment unchanged for shift invariant cost functions.

The important step is the denoising. Several projections to lower dimensions have been tested and $t = 5$ turned out to be a good choice, eliminating the bulk of noise while retaining the essential cluster structure.

Figure 1 shows the striking improvement of the distance matrix after denoising. On the left hand side the ideal distance matrix is depicted, consisting solely of 0's (black) and 1's (white), reflecting the true cluster membership. In the middle and on the right the original and the denoised distance matrix are shown, respectively. Denoising visibly accentuates the cluster structure in the pairwise data. Since we

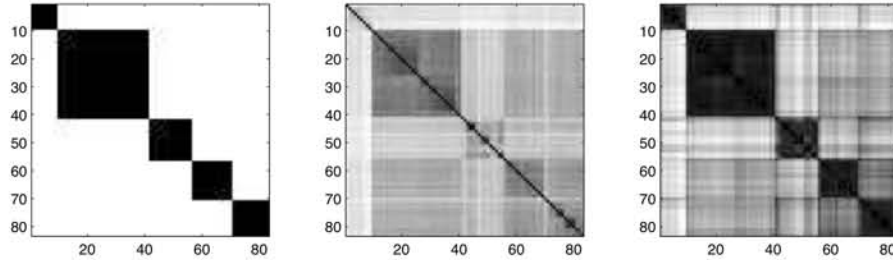

Figure 1: Distance matrix: On the left the ideal distance matrix reflects the true cluster structure. In the middle and on the right: distance matrix before and after denoising

dispose of the true labels, we can quantitatively assess the improvement by denoising. We performed usual $k$-means clustering, followed by a majority voting to match cluster labeling. For the denoised data we obtained 3 misclassifications (3.61%) whereas we got 17 (20.48%) for the original data. This simple experiment corroborates the usefulness of our embedding and denoising strategy for pairwise data.

In order to fulfill the spirit of the theory of constant-shift embedding, the cost-function of the data-mining algorithm subsequent to the embedding needs to be shift invariant. We may by the same token go a step further and apply algorithms for which this condition does not hold. In doing so, however, we give up the mathematical traceability of the error.

To illustrate that denoised pairwise data can act as standalone quality data independent of the framework of algorithms based on shift invariant cost functions (and in order to compare to the results obtained in [18]), a linear SVM is trained on 25% of the total data to mutually classify the genera-pairs: $3 - 4$, $3 - 5$, $4 - 5$. Genera 1 and 2 separate errorless and have therefore been omitted. Model selection over the regularization parameter $C$ has been performed by choosing the optimal value out of 10 equally spaced values from $[10^{-4}, 10^2]$. The results and have been averaged by a 1000-fold sampling (cf. table 1). The best values are printed in bold.

For the classification of genera $3 - 5$ and $4 - 5$ we obtain a substantial improvement by denoising. Interestingly this is not the case for genera $3 - 4$ which may be due to the elimination of discriminative features by the denoising procedure. The error still is significantly smaller than the error obtained by MCK2 and FK, which is in agreement with the superiority of a structure preserving embedding of Smith-Waterman scores even when left undenoised: FK and MCK are kernels de-

| Genera | FK | MCK2 | Undenoised | Denoised |
|---|---|---|---|---|
| 3 − 4 | 10.4 | 8.48 | **5.06** | 5.43 |
| 3 − 5 | 10.9 | 5.71 | 5.72 | **3.83** |
| 4 − 5 | 23.1 | 11.6 | 7.55 | **3.17** |

Table 1: Comparison of mean test-error of supervised classification by linear SVM of genera with training sample 25 % of the total sample. The results for MCK2 (Marginalized Count Kernel) and FK (Fisher Kernel) is obtained by kernel Fisher discriminant analysis which compares favorably to the SVM in several benchmarks [18].

rived from a generative model, whereas the alignment scores are obtained from a matching algorithm specifically tuned for protein sequences, reflecting much better the underlying structure of protein data.

## 3.2 Clustering of ProDom sequences

The analysis described in this section aims at finding a partition of domain sequences from the ProDom database, [3], that is meaningful w.r.t. *structural similarity*. In order to measure the quality of the grouping solution, we use the computed solution in a predictive way to assign group labels to SCOP sequences, which have been labeled by experts according to their structure, [10]. The predicted labels are then compared with the "true" SCOP labels.

For demonstration purposes, we select the following subset of sequences from `prodom2001.2.srs`: among all sequences we choose those which are highly similar to at least one sequence contained in the first four folds of the SCOP database.[2] Between these sequences, we compute pairwise (length-corrected and standardized) Smith-Waterman alignment scores, summarized in the matrix $(S_{ij})$. These similarities are transformed into dissimilarities by setting $D_{ij} := S_{ii} + S_{jj} - 2S_{ij}$. The centralized score matrix $S^c = -1/2D^c$ possesses some highly negative eigenvalues, indicating that metric properties are violated. Applying the constant-shift embedding method, a valid Mercer kernel is derived, with an eigenvalue spectrum that shows only a few dominating components over a broad "noise"-spectrum (see figure 2). Extracting the first 16 leading principal components[3] leads to a vector representation of the sequences as points in $\mathbb{R}^{16}$. These points are then clustered by minimizing the $k$-means cost function within a deterministic annealing framework. The model order was selected by applying a re-sampling based *stability* analysis, which has been demonstrated to be a suitable model order selection criterion for unsupervised grouping problems in [13].

In order to measure the quality of the grouping solution, all 1158 SCOP sequences from the first four folds are embedded into the 16-dimensional space. The predicted group structure on this test set is then compared with the true SCOP fold-labels. Figure 3 shows both the predicted group membership of these sequences and their true SCOP fold-label in the form of a bar diagram: the sequences are ordered by increasing group label (the lower horizontal bar), and compared with the true fold classification (upper bar). In order to quantify the results, the inferred clusters are

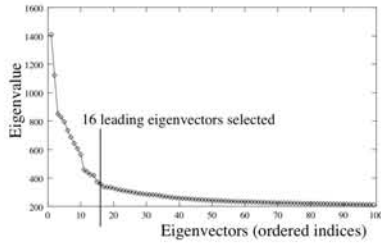

Figure 2: (Partial) eigenvalue spectrum of the shifted score matrix. The data are projected onto the first leading 16 eigenvectors, whereas the remaining principal components are considered to be dominated by noise.

re-labeled ("re-colored") according to the maximum number of correctly identifiable fold-labels. This procedure allows us to correctly identify the fold label of roughly 94 % of the SCOP sequences.

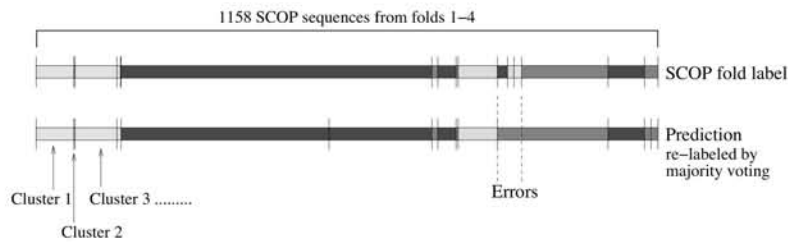

Figure 3: Visualization of cluster membership of the SCOP sequences contained in folds 1-4.

Despite this surprisingly high percentage, it is necessary to deeper analyze the biological relevance of the inferred grouping solution. In order to check to what extent the above "over-all" result is influenced by artefacts due to highly related (or even almost identical) SCOP sequences, we repeated the analysis based on the subset of 128 SCOP sequences with less than 50 % sequence identity (PDB-50). Predicting the group membership of these 128 sequences and using the same re-labeling approach, we can correctly identify 86 % of the fold-labels. This result demonstrates that we have not only found trivial groups of almost identical proteins, but that we have indeed extracted relevant structural information.

## 4 Discussion and Conclusion

This paper provides two main contributions that are highly useful when analyzing pairwise data. First, we employ the concept of constant shift embedding to provide a metric representation of the data. For a certain class of grouping principles sharing a shift-invariance property, this embedding is *distortion-less* in the sense that it does not influence the optimal assignments of objects to groups. Given the metricized data we can now use common signal (pre-)processing and denoising techniques that are typically only defined for vectorial data.

As we investigate the clustering of protein sequences from data bases like GyrB and ProDom, we are given non-metric pairwise proximity information that is strongly deteriorated by the shortcomings of the available alignment procedures. Thus, it is important to apply denoising techniques to the data as a second step before running the actual clustering procedure. We find that the combination of these two processing steps is successful in unraveling protein structure, greatly improving over existing methods (as exemplified for GyrB and ProDom).

Future research will be dedicated to further evaluation of the proposed algorithm. We will also explore the perspectives it opens in any field handling pairwise data.

**Acknowledgments** The *gyrB* amino acid sequences where offered by courtesy of Identification and Classification of Bacteria (ICB) databank team [19]. The authors are partially supported by DFG grants # MU 987/1-1 and # BU 914/4-1.

## Footnotes

[1]The term shift-invariance means that the optimal assignments of objects to clusters are not influenced by constant additive shifts of the pairwise dissimilarities (excluding the self-dissimilarities which are assumed to be zero).

[2]"Highly similar" here means that the highest alignment score exceeds a predefined threshold. The result is a subset of roughly 2700 ProDom domain sequences.

[3]Subsampling techniques or deflation can be used to reduce computational load for large-scale problems. We only used a subset of 800 randomly chosen proteins for estimating the 16 leading eigenvectors.

# References

[1] A.K.Jain, M.N. Murty, and P.J. Flynn. Data clustering: a review. *ACM Computing Surveys*, 31(3):264–323, 1999.

[2] S. F. Altschul, W. Gish, W. Miller, E. W. Myers, and D. J. Lipman. Basic local alignment search tool. *J. Mol. Biol.*, 215:403 – 410, 1990.

[3] F. Corpet, F. Servant, J. Gouzy, and D. Kahn. Prodom and prodom-cg: tools for protein domain analysis and whole genome comparisons. *Nucleid Acids Res.*, 28:267–269, 2000.

[4] T. F. Cox and M. A. A. Cox. *Multidimensional Scaling*. Chapman & Hall, London, 2001.

[5] R.O. Duda, P.E.Hart, and D.G.Stork. *Pattern classification*. John Wiley & Sons, second edition, 2001.

[6] P. J. Huber. Projection pursuit. *The Annals of Statistics*, pages 435–475, 1985.

[7] H. Kasai, A. Bairoch, K. Watanabe, K. Isono, and S. Harayama. Construction of the gyrb database for the identification and classification of bacteria. *Genome Informatics*, pages 13 – 21, 1998.

[8] T. Kohonen. *Self-Organizing Maps*. Springer-Verlag, Berlin, 1995.

[9] S. Mika, B. Schölkopf, A.J. Smola, K.-R. Müller, M. Scholz, and G. Rätsch. Kernel PCA and de–noising in feature spaces. In M.S. Kearns, S.A. Solla, and D.A. Cohn, editors, *Advances in Neural Information Processing Systems*, volume 11, pages 536–542. MIT Press, 1999.

[10] A.G. Murzin, S.E. Brenner, T. Hubbard, and C. Chothia. Scop: a structural classification of proteins database for the investigation of sequences and structures. *J. Mol. Biol.*, 247:536–540, 1995.

[11] W. R. Pearson and D. J. Lipman. Improved tools for biological sequence analysis. *Proc. Natl. Acad. Sci*, 85:2444 – 2448, 1988.

[12] J. Puzicha, T. Hofmann, and J. Buhmann. A theory of proximity based clustering: Structure detection by optimization. *Pattern Recognition*, 33(4):617–634, 1999.

[13] V. Roth, M. Braun, T. Lange, and J. Buhmann. A resampling approach to cluster validation. In *Computational Statistics–COMPSTAT'02*, 2002. To appear.

[14] V. Roth, J. Laub, M. Kawanabe, and J.M. Buhmann. Optimal cluster preserving embedding of non-metric proximity data. Technical Report IAI-TR-2002-5, University of Bonn, 2002.

[15] S. Roweis and L. Saul. Nonlinear dimensionality reduction by locally linear embedding. *Science*, 290:2323–2326, 2000.

[16] B. Schölkopf, A. Smola, and K.-R. Müller. Nonlinear component analysis as a kernel eigenvalue problem. *Neural Computation*, 10:1299–1319, 1998.

[17] J.B. Tenenbaum, V. Silva, and J.C. Langford. A global geometric framework for nonlinear dimensionality reduction. *Science*, 290:2319–2323, 2000.

[18] K. Tsuda, T. Kin, and K. Asai. Marginalized kernels for biological sequences. *Proc. ISMB*, to appear:2002, http://www.cbrc.jp/ tsuda/.

[19] K. Watanabe, J. Nelson, S. Harayama, and H. Kasai. Icb database: the gyrb database for identification and classification of bacteria. *Nucleic Acids Res.*, 29:344 – 345, 2001.
